# Concurrent Object Recognition and Segmentation by Graph Partitioning

**Stella X. Yu**[†‡]**, Ralph Gross**[†] **and Jianbo Shi**[†]
Robotics Institute[†]
Carnegie Mellon University
Center for the Neural Basis of Cognition[‡]
5000 Forbes Ave, Pittsburgh, PA 15213-3890
{stella.yu, rgross, jshi}@cs.cmu.edu

## Abstract

Segmentation and recognition have long been treated as two separate processes. We propose a mechanism based on spectral graph partitioning that readily combine the two processes into one. A part-based recognition system detects object patches, supplies their partial segmentations as well as knowledge about the spatial configurations of the object. The goal of patch grouping is to find a set of patches that conform best to the object configuration, while the goal of pixel grouping is to find a set of pixels that have the best low-level feature similarity. Through pixel-patch interactions and between-patch competition encoded in the solution space, these two processes are realized in one joint optimization problem. The globally optimal partition is obtained by solving a constrained eigenvalue problem. We demonstrate that the resulting object segmentation eliminates false positives for the part detection, while overcoming occlusion and weak contours for the low-level edge detection.

## 1 Introduction

A good image segmentation must single out meaningful structures such as objects from a cluttered scene. Most current segmentation techniques take a bottom-up approach [5], where image properties such as feature similarity (brightness, texture, motion etc), boundary smoothness and continuity are used to detect perceptually coherent units. Segmentation can also be performed in a top-down manner from object models, where object templates are projected onto an image and matching errors are used to determine the existence of the object [1]. Unfortunately, either approach alone has its drawbacks.

Without utilizing any knowledge about the scene, image segmentation gets lost in poor data conditions: weak edges, shadows, occlusions and noise. Missed object boundaries can then hardly be recovered in subsequent object recognition. Gestaltists have long recognized this issue, circumventing it by adding a grouping factor called *familiarity* [6]. Without being subject to perceptual constraints imposed by low level grouping, an object detection process can produce many false positives in a cluttered scene [3]. One approach is to build a better part detector, but this has its own limitations, such as increase in the complexity of classifiers and the number of training examples required. Another approach, which we adopt in

this paper, is based on the observation that the falsely detected parts are not perceptually salient (Fig. 1), thus they can be effectively pruned away by perceptual organization.

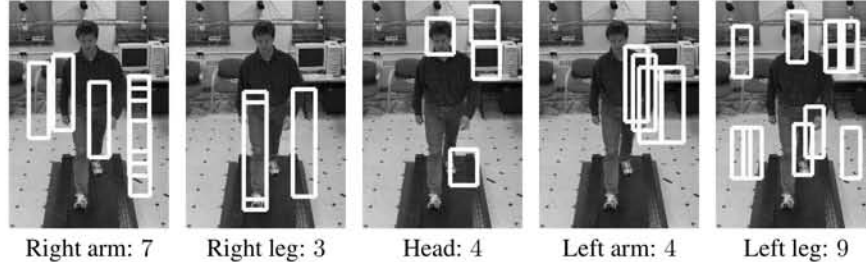

| Right arm: 7 | Right leg: 3 | Head: 4 | Left arm: 4 | Left leg: 9 |

Figure 1: Human body part detection. A total of 27 parts are detected, each labeled by one of the five part detectors for arms, legs and head. False positives cannot be validated on two grounds. First, they do not form salient structures based on low-level cues, e.g. the patch on the floor that is labeled left leg has the same features as its surroundings. Secondly, false positives are often incompatible with nearby parts, e.g. the patch on the treadmill that is labeled head has no other patches in the image to make up a whole human body. These two conditions, low-level image feature saliency and high-level part labeling consistency, are essential for the segmentation of objects from background. Both cues are encoded in our pixel and patch grouping respectively.

We propose a segmentation mechanism that is coupled with the object recognition process (Fig. 2). There are three tightly coupled processes. 1)Top-level: part-based object recognition process. It learns classifiers from training images to detect parts along with the segmentation patterns and their relative spatial configurations. A few approaches based on pattern classification have been developed for part detection [9, 3]. Recent work on object segmentation [1] uses image patches and their figure-ground labeling as building blocks for segmentation. 2)Bottom-level: pixel-based segmentation process. This process finds perceptually coherent groups using pairwise local feature similarity. The local features we use here are contour cues. 3)Interactions: coupling object recognition with segmentation by linking patches with their corresponding pixels. With such a representation, we concurrently carry out object recognition and image segmentation processes. The final output is an object segmentation where the object group consists of pixels with coherent low-level features and patches with compatible part configurations.

We formulate our object segmentation task in a graph partitioning framework. We represent low-level grouping cues with a graph where each pixel is a node and edges between the nodes encode the affinity of pixels based on their feature similarity [4]. We represent high-level grouping cues with a graph where each detected patch is a node and edges between the nodes encode the labeling consistency based on prior knowledge of object part configurations. There are also edges connecting patch nodes with their supporting pixel nodes. We seek the optimal graph cut in this joint graph, which separates the desired patch and pixel nodes from the rest nodes. We build upon the computational framework of spectral graph partitioning [7], and achieve patch competition using the subspace constraint method proposed in [10]. We show that our formulation leads to a constrained eigenvalue problem, whose global-optimal solutions can be obtained efficiently.

## 2   Segmentation model

We illustrate our method through a synthetic example shown in Fig. 3. Suppose we are interested in detecting a human-like configuration. Furthermore, we assume that some object recognition system has labeled a set of patches as object parts. Every patch has a local segmentation according to its part label. The recognition system has also learned the

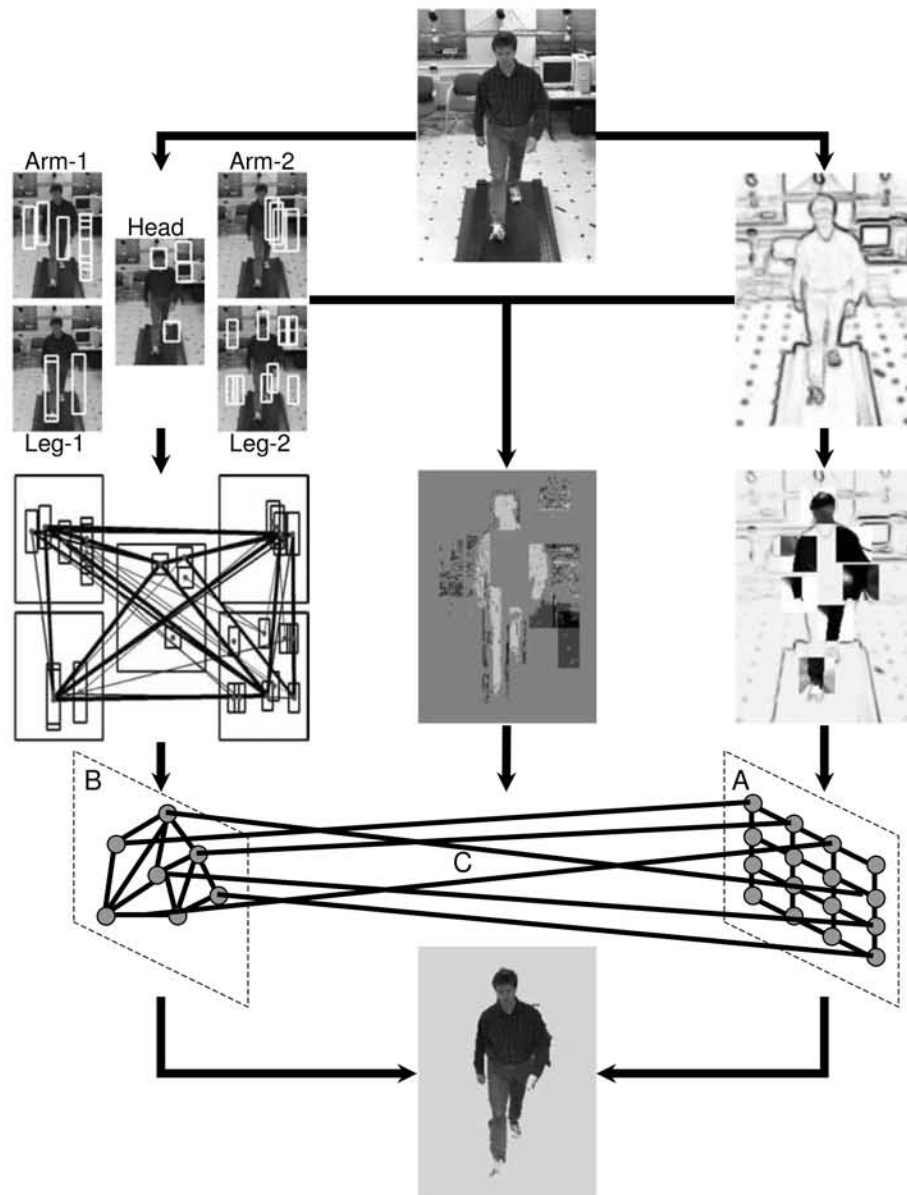

Figure 2: Model of object segmentation. Given an image, we detect edges using a set of oriented filter banks. The edge responses provide low-level grouping cues, and a graph can be constructed with one node for each pixel. Shown on the middle right are affinity patterns of five center pixels within a square neighbourhood, overlaid on the edge map. Dark means larger affinity. We detect a set of candidate body parts using learned classifiers. Body part labeling provides high-level grouping cues, and a consistency graph can be constructed with one node for each patch. Shown on the middle left are the connections between patches. Thicker lines mean better compatibility. Edges are noisy, while patches contain ambiguity in local segmentation and part labeling. Patches and pixels interact by expected local segmentation based on object knowledge, as shown in the middle image. A global partitioning on the coupled graph outputs an object segmentation that has both pixel-level saliency and patch-level consistency.

statistical distribution of the spatial configurations of object parts. Given such information, we need to address two issues. One is the cue evaluation problem, i.e. how to evaluate low-level pixel cues, high-level patch cues and their segmentation correspondence. The other is the integration problem, i.e. how to fuse partial and imprecise object knowledge with somewhat unreliable low-level cues to segment out the object of interest.

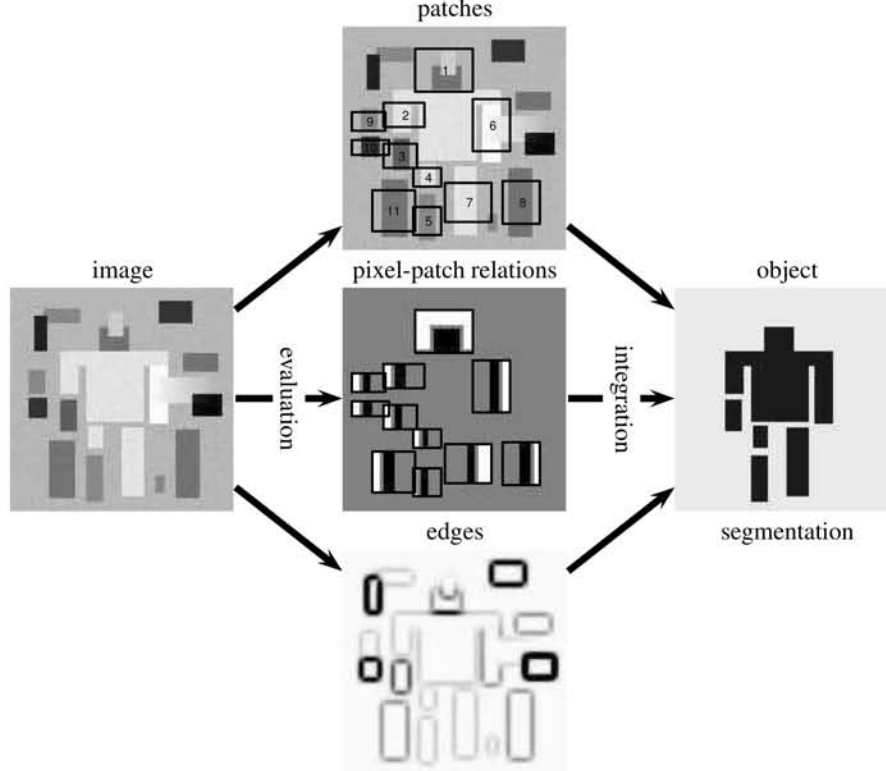

Figure 3: Given the image on the left, we want to detect the object on the right). 11 patches of various sizes are detected (middle top). They are labeled as head(1), left-upper-arm(2, 9), left-lower-arm(3, 10), left-leg (11), left-upper-leg(4), left-lower-leg(5), right-arm(6), right-leg(7, 8). Each patch has a *partial* local segmentation as shown in the center image. Object pixels are marked black, background white and others gray. The image intensity itself has its natural organization, e.g. pixels across a strong edge (middle bottom) are likely to be in different regions. Our goal is to find the best patch-pixel combinations that conform to the object knowledge and data coherence.

## 2.1 Representations

We denote the graph in Fig. 2 by $\mathsf{G} = (\mathsf{V}, \mathsf{E}, W)$. Let $N$ be the number of pixels and $M$ the number of patches. Let $A$ be the pixel-pixel affinity matrix, $B$ be the patch-patch affinity matrix, and $C$ be the patch-pixel affinity matrix. All these weights are assumed *nonnegative*. Let $\beta_B$ and $\beta_C$ be scalars reflecting the relative importance of $B$ and $C$ with respect to $A$. Then the node set and the weight matrix for the pairwise edge set $\mathsf{E}$ are:

$$\mathsf{V} = \{ \underbrace{1, \cdots, N,}_{\text{pixels}} \underbrace{N+1, \cdots, N+M}_{\text{patches}} \},$$

$$W(A, B, C; \beta_B, \beta_C) = \begin{bmatrix} A_{N \times N} & \beta_C \cdot C^T_{N \times M} \\ \beta_C \cdot C_{M \times N} & \beta_B \cdot B_{M \times M} \end{bmatrix}. \tag{1}$$

Object segmentation corresponds to a node bipartitioning problem, where $V = V_1 \cup V_2$ and $V_1 \cap V_2 = \varnothing$. We assume $V_1$ contains a set of pixel and patch nodes that correspond to the object, and $V_2$ is the rest of the background pixels and patches that correspond to false positives and alternative labelings. Let $X_1$ be an $(N + M) \times 1$ vector, with $X_1(k) = 1$ if node $k \in V_1$ and 0 otherwise. It is convenient to introduce the indicator for $V_2$, where $X_2 = 1 - X_1$ and 1 is the vector of ones.

We only need to process the image region enclosing all the detected patches. The rest pixels are associated with a virtual background patch, which we denote as patch $N + M$, in addition to $M - 1$ detected object patches. Restriction of segmentation to this region of interest (ROI) helps binding irrelavent background elements into one group [10].

## 2.2 Computing pixel-pixel similarity $A$

The pixel affinity matrix $A$ measures low-level image feature similarity. In this paper, we choose intensity as our feature and calcuate $A$ based on edge detection results. We first convolve the image with quadrature pairs of oriented filters to extract the magnitude of edge responses $OE$ [4]. Let $\underline{i}$ denote the location of pixel $i$. Pixel affinity $A$ is inversely correlated with the maximum magnitude of edges crossing the line connecting two pixels. $A(i, j)$ is low if $i, j$ are on the two sides of a strong edge (Fig. 4):

$$A(i, j) = \exp\left(-\frac{1}{2\sigma_e^2} \cdot \left[\frac{\max_{t \in (0,1)} OE(\underline{i} + t \cdot \underline{j})}{\max_k OE(\underline{k})}\right]^2\right). \tag{2}$$

$$A(1, 3) \approx 1$$
$$A(1, 2) \approx 0$$

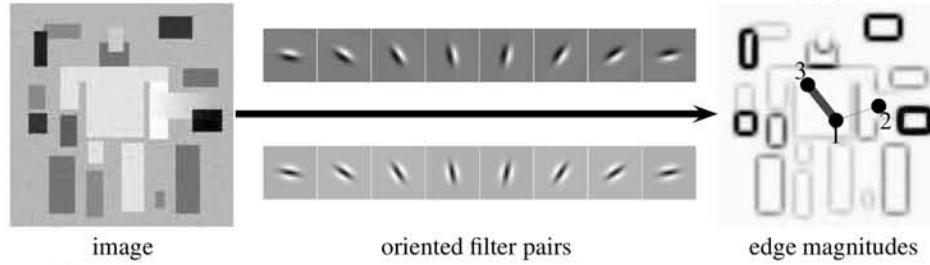

image        oriented filter pairs        edge magnitudes

Figure 4: Pixel-pixel similarity matrix $A$ is computed based on intensity edge magnitudes.

## 2.3 Computing patch-patch compatibility $B$ and competition

For object patches, we evaluate their position compatibility according to learned statistical distributions. For object part labels $a$ and $b$, we can model their spatial distribution by a Gaussian, with mean $\mu_{ab}$ and variance $\Sigma_{ab}$ estimated from training data. Let $\acute{p}$ be the object label of patch $p$. Let $\underline{p}$ be the center location of patch $p$. For patches $p$ and $q$, $B(p, q)$ is low if $p, q$ form rare configurations for their part labels $\acute{p}$ and $\acute{q}$ (Fig. 5a):

$$B(p, q) = \exp\left(-\frac{1}{2}(\underline{p} - \underline{q} - \mu_{\acute{p}\acute{q}})^T \Sigma_{\acute{p}\acute{q}}^{-1}(\underline{p} - \underline{q} - \mu_{\acute{p}\acute{q}})\right). \tag{3}$$

We manually set these values for our image examples. As to the virtual background patch node, it only has affinity of 1 to itself.

Patch compatibility measures alone do not prevent the desired pixel and patch group from including falsely detected patches and their pixels, nor does it favor the true object pixels to be away from unlabeled background pixels. We need further constraints to restrict a feasible grouping. This is done by constraining the partition indicator $X$. In Fig. 5b, there are four pairs of patches with the same object part labels. To encode mutual exclusion between patches, we enforce one winner among patch nodes in competition. For example, only one of the patches 7 and 8 can be validated to the object group: $X_1(N + 7) + X_1(N + 8) = 1$.

We also set an exclusion constraint between a reliable patch and the virtual background patch so that the desired object group stands out alone without these unlabeled background pixels, e.g $X_1(N+1) + X_1(N+M) = 1$. Formally, let $S$ be a superset of nodes to be separated and let $|\cdot|$ denote the cardinality of a set. We have:

$$\sum_{k \in S_m} X_1(k) = 1, \ m = 1 : |S|. \tag{4}$$

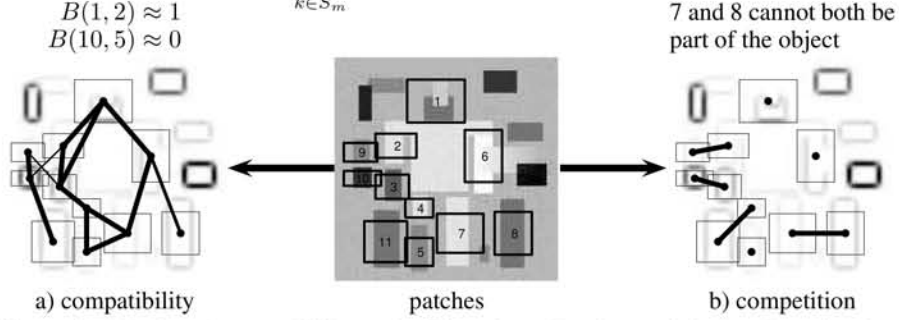

$B(1,2) \approx 1$
$B(10,5) \approx 0$

7 and 8 cannot both be part of the object

a) compatibility        patches        b) competition

Figure 5: a) Patch-patch compatibility matrix $B$ is evaluated based on statistical configuration plausibility. Thicker lines for larger affinity. b) Patches of the same object part label compete to enter the object group. Only one winner from each linked pair of patches can be validated as part of the object.

### 2.4 Computing pixel-patch association $C$

Every object part label also projects an expected pixel segmentation within the patch window (Fig. 6). The pixel-patch association matrix $C$ has one column for each patch:

$$C(i,p) = \begin{cases} 1, & \text{if } i \text{ is an object pixel of patch } p, \\ 0, & \text{otherwise.} \end{cases} \tag{5}$$

For the virtual background patch, its member pixels are those outside the ROI.

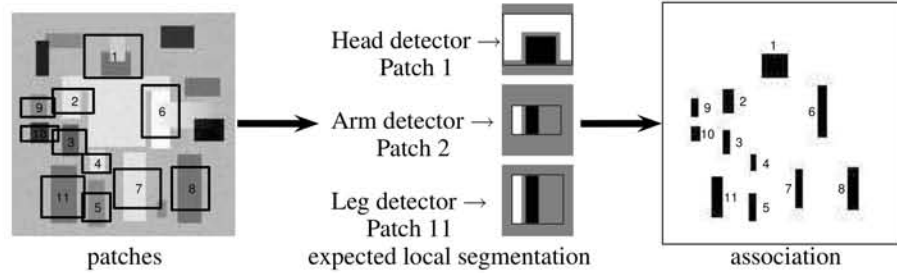

Head detector →
Patch 1

Arm detector →
Patch 2

Leg detector →
Patch 11

patches        expected local segmentation        association

Figure 6: Pixel-patch association $C$ for object patches. Object pixels are marked black, background white and others gray. A patch is associated with its object pixels in the given *partial* segmentation.

Finally, we desire $\beta_B$ to balance the total weights between pixel and patch grouping so that $M \ll N$ does not render patch grouping insignificant, and we want $\beta_C$ to be large enough so that the results of patch grouping can bring along their associated pixels:

$$\beta_B = 0.01 \frac{1^T A 1}{1^T B 1}, \qquad \beta_C = \frac{\beta_B}{\max C}. \tag{6}$$

### 2.5 Segmentation as an optimization problem

We apply the normalized cuts criterion [7] to the joint pixel-patch graph in Eq. (1):

$$\max \epsilon(X_1) = \sum_{t=1}^{2} \frac{X_t^T W X_t}{X_t^T D X_t}, \quad \text{s. t.} \sum_{k \in S_m} X_1(k) = 1, \ m = 1 : |S|. \tag{7}$$

$D$ is the diagonal degree matrix of $W$, $D(i,i) = \sum_j W(i,j)$. Let $x = X_1 - \frac{X_1^T D X_1}{1^T D 1}$. By relaxing the constraints into the form of $L^T x = 0$ [10], Eq. (7) becomes a constrained eigenvalue problem [10], the maximizer given by the nontrivial leading eigenvector:

$$x^* = \arg\max \frac{x^T W x}{x^T D x}, \quad \text{s. t. } L^T x = 0. \tag{8}$$

$$QD^{-1}Wx^* = \lambda x^*, \tag{9}$$

$$Q = I - D^{-1}L(L^T D^{-1}L)^{-1}L^T. \tag{10}$$

Once we get the optimal eigenvector, we compare 10 thresholds uniformly distributed within its range and choose the discrete segmentation that yields the best criterion $\epsilon$. Below is an overview of our algorithm.

1: Compute edge response $OE$ and calculate pixel affinity $A$, Eq. (2).
2: Detect parts and calculate patch affinity $B$, Eq. (3).
3: Formulate constraints $S$ and $L$ among competing patches, Eq. (4).
4: Set pixel-patch affinity $C$, Eq. (5).
5: Calculate weights $\beta_B$ and $\beta_C$, Eq. (6).
6: Form $W$ and calculate its degree matrix $D$, Eq. (1).
7: Solve $QD^{-1}Wx^* = \lambda x^*$, Eq. (9,10).
8: Threshold $x^*$ to get a discrete segmentation.

## 3 Results and conclusions

In Fig. 7, we show results on the $120 \times 120$ synthetic image. Image segmentation alone gets lost in a cluttered scene. With concurrent segmentation and recognition, regions forming the object of interest pop out, with unwanted edges (caused by occlusion) and weak edges (illusory contours) corrected in the final segmentation. It is also faster to compute the pixel-patch grouping since the size of the solution space is greatly reduced.

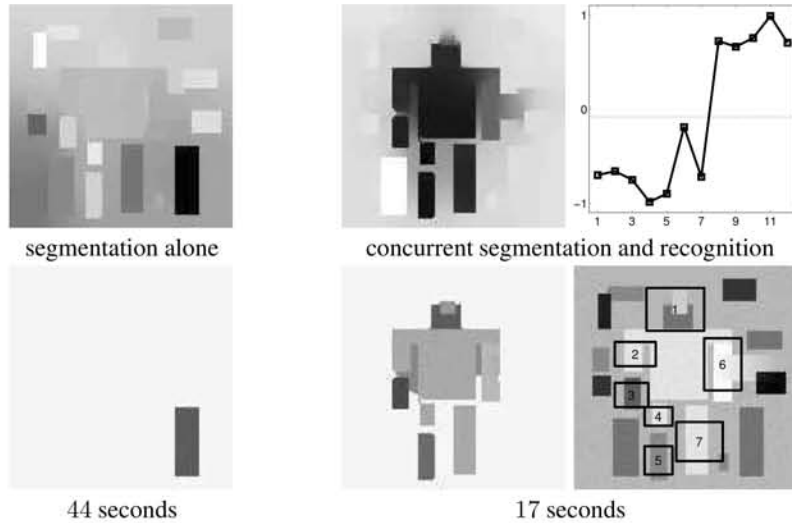

segmentation alone     concurrent segmentation and recognition

44 seconds     17 seconds

Figure 7: Eigenvectors (row 1) and their segmentations (row 2) for Fig. 3. On the right, we show the optimal eigenvector on both pixels and patches, the horizontal dotted line indicating the threshold. Computation times are obtained in MATLAB 6.0 on a PC with 1GHz CPU and 1G memory.

We apply our method to human body detection in a single image. We manually label five body parts (both arms, both legs and the head) of a person walking on a treadmill in all

32 images of a complete gait cycle. Using the magnitude thresholded edge orientations in the hand-labeled boxes as features, we train linear Fisher classifiers [2] for each body part. In order to account for the appearance changes of the limbs through the gait cycle, we use two separate models for each arm and each leg, bringing the total number of models to 9. Each individual classifier is trained to discriminate between the body part and a random image patch. We iteratively re-train the classifiers using false positives until the optimal performance is reached over the training set. In addition, we train linear color-based classifiers for each body part to perform figure-ground discrimination at the pixel level. Alternatively a general model of human appearance based on filter responses as in [8] could be used. In Fig. 8, we show the results on the test image in Fig. 2. Though the pixel-patch affinity matrix $C$, derived from the color classifier, is neither precise nor complete, and the edges are weak at many object boundaries, the two processes complement each other in our pixel-patch grouping system and output a reasonably good object segmentation.

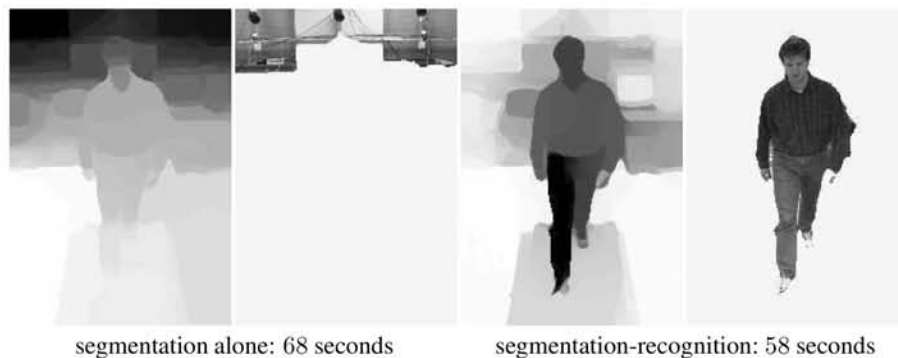

segmentation alone: 68 seconds      segmentation-recognition: 58 seconds

Figure 8: Eigenvectors and their segmentations for the $261 \times 183$ human body image in Fig. 2.

**Acknowledgments**. We thank Shyjan Mahamud and anonymous referees for valuable comments. This research is supported by ONR N00014-00-1-0915 and NSF IRI-9817496.

# References

[1] E. Borenstein and S. Ullman. Class-specific, top-down segmentation. In *European Conference on Computer Vision*, 2002.

[2] K. Fukunaga. *Introduction to statistical pattern recognition*. Academic Press, 1990.

[3] S. Mahamud, M. Hebert, and J. Lafferty. Combining simple discriminators for object discrimination. In *European Conference on Computer Vision*, 2002.

[4] J. Malik, S. Belongie, T. Leung, and J. Shi. Contour and texture analysis for image segmentation. *International Journal of Computer Vision*, 2001.

[5] D. Marr. *Vision*. CA: Freeman, 1982.

[6] S. E. Palmer. *Vision science: from photons to phenomenology*. MIT Press, 1999.

[7] J. Shi and J. Malik. Normalized cuts and image segmentation. In *IEEE Conference on Computer Vision and Pattern Recognition*, pages 731–7, June 1997.

[8] H. Sidenbladh and M. Black. Learning image statistics for Bayesian tracking. In *International Conference on Computer Vision*, 2001.

[9] P. Viola and M. Jones. Rapid object detection using a boosted cascade of simple features. In *IEEE Conference on Computer Vision and Pattern Recognition*, 2001.

[10] S. X. Yu and J. Shi. Grouping with bias. In *Neural Information Processing Systems*, 2001.
